# Modeling image patches with a directed hierarchy of Markov random fields

**Simon Osindero and Geoffrey Hinton**
Department of Computer Science, University of Toronto
6, King's College Road, M5S 3G4, Canada
osindero,hinton@cs.toronto.edu

## Abstract

We describe an efficient learning procedure for multilayer generative models that combine the best aspects of Markov random fields and deep, directed belief nets. The generative models can be learned one layer at a time and when learning is complete they have a very fast inference procedure for computing a good approximation to the posterior distribution in all of the hidden layers. Each hidden layer has its own MRF whose energy function is modulated by the top-down directed connections from the layer above. To generate from the model, each layer in turn must settle to equilibrium given its top-down input. We show that this type of model is good at capturing the statistics of patches of natural images.

## 1  Introduction

The soldiers on a parade ground form a neat rectangle by interacting with their neighbors. An officer decides where the rectangle should be, but he would be ill-advised to try to tell each individual soldier exactly where to stand. By allowing constraints to be enforced by local interactions, the officer enormously reduces the bandwidth of top-down communication required to generate a familiar pattern. Instead of micro-managing the soldiers, the officer specifies an objective function and leaves it to the soldiers to optimise that function. This example of pattern generation suggests that a multilayer, directed belief net may not be the most effective way to generate patterns. Instead of using shared ancestors to create correlations between the variables within a layer, it may be more efficient for each layer to have its own energy function that is modulated by directed, top-down input from the layer above. Given the top-down input, each layer can then use lateral interactions to settle on a good configuration and this configuration can then provide the top-down input for the next layer down. When generating an image of a face, for example, the approximate locations of the mouth and nose might be specified by a higher level and the local interactions would then ensure that the accuracy of their vertical alignment was far greater than the accuracy with which their locations were specified top-down.

In this paper, we show that recently developed techniques for learning deep belief nets (DBN's) can be generalized to solve the apparently more difficult problem of learning a directed hierarchy of Markov Random Fields (MRF's). The method we describe can learn models that have many hidden layers, each with its own MRF whose energy function is conditional on the values of the variables in the layer above. It does not require detailed prior knowledge about the data to be modeled, though it obviously works better if the architecture and the types of latent variable are well matched to the task.

## 2  Learning deep belief nets: An overview

The learning procedure for deep belief nets has now been described in several places (Hinton et al., 2006; Hinton and Salakhutdinov, 2006; Bengio et al., 2007) and will only be sketched here. It relies on a basic module, called a restricted Boltzmann machine (RBM) that can be trained efficiently using a method called "contrastive divergence" (Hinton, 2002).

### 2.1  Restricted Boltzmann Machines

An RBM consists of a layer of binary stochastic "visible" units connected to a layer of binary, stochastic "hidden" units via symmetrically weighted connections. A joint configuration, $(\mathbf{v}, \mathbf{h})$ of the visible and hidden units has an energy given by:

$$E(\mathbf{v}, \mathbf{h}) = - \sum_{i \in \text{visibles}} b_i v_i - \sum_{j \in \text{hiddens}} b_j h_j - \sum_{i,j} v_i h_j w_{ij} \tag{1}$$

where $v_i, h_j$ are the binary states of visible unit $i$ and hidden unit $j$, $b_i, b_j$ are their biases and $w_{ij}$ is the symmetric weight between them. The network assigns a probability to every possible image via this energy function and the probability of a training image can be raised by adjusting the weights and biases to lower the energy of that image and to raise the energy of similar, reconstructed images that the network would prefer to the real data.

Given a training vector, $\mathbf{v}$, the binary state, $h_j$, of each feature detector, $j$, is set to 1 with probability $\sigma(b_j + \sum_i v_i w_{ij})$, where $\sigma(x)$ is the logistic function $1/(1 + \exp(-x))$, $b_j$ is the bias of $j$, $v_i$ is the state of visible unit $i$, and $w_{ij}$ is the weight between $i$ and $j$. Once binary states have been chosen for the hidden units, a reconstruction is produced by setting each $v_i$ to 1 with probability $\sigma(b_i + \sum_j h_j w_{ij})$. The states of the hidden units are then updated once more so that they represent features of the reconstruction. The change in a weight is given by

$$\Delta w_{ij} = \epsilon(\langle v_i h_j \rangle_{data} - \langle v_i h_j \rangle_{recon}) \tag{2}$$

where $\epsilon$ is a learning rate, $\langle v_i h_j \rangle_{data}$ is the fraction of times that visible unit $i$ and hidden units $j$ are on together when the hidden units are being driven by data and $\langle v_i h_j \rangle_{recon}$ is the corresponding fraction for reconstructions. A simplified version of the same learning rule is used for the biases. The learning works well even though it is not exactly following the gradient of the log probability of the training data (Hinton, 2002).

### 2.2  Compositions of experts

A single layer of binary features is usually not the best way to capture the structure in the data. We now show how RBM'S can be composed to create much more powerful, multilayer models.

After using an RBM to learn the first layer of hidden features we have an undirected model that defines $p(\mathbf{v}, \mathbf{h})$ via the energy function in Eq. 1. We can also think of the model as defining $p(\mathbf{v}, \mathbf{h})$ by defining a consistent pair of conditional probabilities, $p(\mathbf{h}|\mathbf{v})$ and $p(\mathbf{v}|\mathbf{h})$ which can be used to sample from the model distribution. A different way to express what has been learned is $p(\mathbf{v}|\mathbf{h})$ and $p(\mathbf{h})$. Unlike a standard directed model, this $p(\mathbf{h})$ does not have its own separate parameters. It is a complicated, non-factorial prior on $\mathbf{h}$ that is defined implicitly by the weights. This peculiar decomposition into $p(\mathbf{h})$ and $p(\mathbf{v}|\mathbf{h})$ suggests a recursive algorithm: keep the learned $p(\mathbf{v}|\mathbf{h})$ but replace $p(\mathbf{h})$ by a better prior over $\mathbf{h}$, *i.e.* a prior that is closer to the average, over all the data vectors, of the conditional posterior over $\mathbf{h}$.

We can sample from this average conditional posterior by simply applying $p(\mathbf{h}|\mathbf{v})$ to the training data. The sampled $\mathbf{h}$ vectors are then the "data" that is used for training a higher-level RBM that learns the next layer of features. We could initialize the higher-level RBM model by using the same parameters as the lower-level RBM but with the roles of the hidden and visible units reversed. This ensures that $p(\mathbf{v})$ for the higher-level RBM starts out being exactly the same as $p(\mathbf{h})$ for the lower-level one. Provided the number of features per layer does not decrease, Hinton et al. (2006) show that each extra layer increases a variational lower bound on the log probability of the data.

The directed connections from the first hidden layer to the visible units in the final, composite graphical model are a consequence of the the fact that we keep the $p(\mathbf{v}|\mathbf{h})$ but throw away the $p(\mathbf{h})$ defined by the first level RBM. In the final composite model, the only undirected connections are

between the top two layers, because we do not throw away the $p(\mathbf{h})$ for the highest-level RBM. To suppress noise in the learning signal, we use the real-valued activation *probabilities* for the visible units of all the higher-level RBM's, but to prevent hidden units from transmitting more than one bit of information from the data to its reconstruction, we always use stochastic binary values for the hidden units.

## 3   Semi-restricted Boltzmann machines

For contrastive divergence learning to work well, it is important for the hidden units to be sampled from their conditional distribution given the data or the reconstructions. It not necessary, however, for the reconstructions to be sampled from their conditional distribution given the hidden states. All that is required is that the reconstructions have lower free energy than the data. So it is possible to include lateral connections between the visible units and to create reconstructions by taking a small step towards the conditional equilibrium distribution given the hidden states. If we are using mean-field activities for the reconstructions, we can move towards the equilibrium distribution by using a few damped mean-field updates (Welling and Hinton, 2002). We call this a semi-restricted Boltzmann machine (SRBM). The visible units form a conditional MRF with the biases of the visible units being determined by the hidden states. The learning procedure for the visible to hidden connections is unaffected and the same learning procedure applies to the lateral connections. Explicitly, the energy function for a SRBM is given by

$$E(\mathbf{v}, \mathbf{h}) = -\sum_{i \in \text{visibles}} b_i v_i - \sum_{j \in \text{hiddens}} b_j h_j - \sum_{i,j} v_i h_j w_{ij} - \sum_{i < i'} v_i v_{i'} L_{ii'} \qquad (3)$$

and the update rule for the lateral connections is

$$\Delta L_{ii'} = \epsilon(\langle v_i v_{i'} \rangle_{data} - \langle v_i v_{i'} \rangle_{recon}) \qquad (4)$$

Semi-restricted Boltzmann machines can be learned greedily and composed to form a directed hierarchy of conditional MRF's. To generate from the composite model we first get an equilbrium sample from the top level SRBM and then we get an equilibrium sample from each lower level MRF in turn, given the top-down input from the sample in the layer above. The settling at each intermediate level does not need to explore a highly multi-modal energy landscape because the top-down input has already selected a good region of the space. The role of the settling is simply to sharpen the somewhat vague top-down specification and to ensure that the resulting configuration repects learned constraints. Each intermediate level fills in the details given the larger picture defined by the level above.

## 4   Inference in a directed hierarchy of MRF's

In a deep belief network, inference is very simple and very fast because of the way in which the network is learned. Rather than first deciding how to represent the data and then worrying about inference afterwards, deep belief nets restrict themselves to learning representations for which accurate variational inference can be done in a single bottom-up pass. Each layer computes an approximate sample from its posterior distribution given the activities in the layer below. This can be done with a single matrix multiply using the bottom-up "recognition" connections that were originally learned by an RBM but are no longer part of the generative model. The recognition connections compute an approximation to the *product* of a data-dependent likelihood term coming from the layer below and a data-independent prior term that depends on the learned parameters of all the higher layers. Each of these two terms can contain strong correlations, but the way in which the model is learned ensures that these correlations cancel each other out so that the true posterior distribution in each layer is very close to factorial and very simple to compute from the activities in the layer below.

The inference process is unaltered by adding an MRF at each hidden layer. The role of the MRF's is to allow the generative process to mimic the constraints that are obeyed by the variables within a layer when the network is being driven bottom-up by data. During inference, these constraints are enforced by the data. From a biological perspective, it is very important for perceptual inference to be fast and accurate, so it is very good that it does not involve any kind of iterative settling or belief propagation. The MRF's are vital for imposing constraints during generation and for whitening the

learning signal so that weak higher-order structure is not masked by strong pairwise correlations. During perceptual inference, however, the MRF's are mere spectators.

## 5    Whitening without waiting

Data is often whitened to prevent strong pairwise correlations from masking weaker but more interesting structure. An alternative to whitening the data is to modify the learning procedure so that it acts *as if* the data were whitened and ignores strong pairwise correlations when learning the next level of features. This has the advantage that perceptual inference is not slowed down by an explicit whitening stage. If the lateral connections ensure that a pairwise correlation in the distribution of the reconstructions is the same as in the data distribution, that correlation will be ignored by contrastive divergence since the learning is driven by the differences between the two distributions. This also explains why different hidden units learn different features even when they have the same connectivity: once one feature has made one aspect of the reconstructions match the data, there is no longer any learning signal for another hidden unit to learn that same aspect.

Figure 1 shows how the features learned by the hidden units are affected by the presence of lateral connections between the visible units. Hidden units are no longer required for modeling the strong pairwise correlations between nearby pixels so they are free to discover more interesting features than the simple on-center off-surround fetaures that are prevalent when there are no connections between visible units.

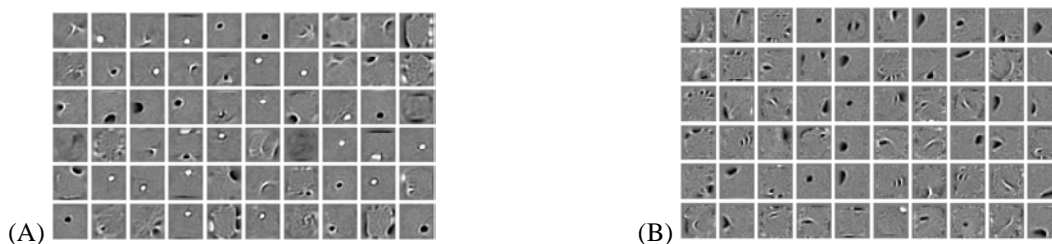

(A)                                (B)

Figure 1: (A) A random sample of the filters learned by an RBM trained on 60,000 images of handwritten digits from the MNIST database (see Hinton et al. (2006) for details). (B) A random sample of the filters learned by an SRBM trained on the same data. To produce each reconstruction, the SRBM used 5 damped mean-field iterations with the top-down input from the hidden states fixed. Adding lateral connections between the visible units changes the types of hidden features that are learned. For simplicity each visible unit in the SRBM was connected to all 783 other visible units, but only the local connections developed large weights and the lateral connections to each pixel formed a small on-center off-surround field centered on the pixel. Pixels close to the edge of the image that were only active one or two times in the whole training set behaved very differently: They learned to predict the whole of the particular digit that caused them to be active.

## 6    Modeling patches of natural images

To illustrate the advantages of adding lateral connections to the hidden layers of a DBN we use the well-studied task of modelling the statistical structure of images of natural scenes (Bell and Sejnowski, 1997; Olshausen and Field, 1996; Karklin and Lewicki, 2005; Osindero et al., 2006; Lyu and Simoncelli, 2006). Using DBN's, it is easy to build overcomplete and hierchical generative models of image patches. These are able to capture much richer types of statistical dependency than traditional generative models such as ICA. They also have the potential to go well beyond the types of dependencies that can be captured by other, more sophisticated, multi-stage approaches such as (Karklin and Lewicki, 2005; Osindero et al., 2006; Lyu and Simoncelli, 2006).

### 6.1    Adapting Restricted Boltzmann machines to real-valued data

Hinton and Salakhutdinov (2006) show how the visible units of an RBM can be modified to allow it to model real-valued data using linear visible variables with Gaussian noise, but retaining the binary stochastic hidden units. The learning procedure is essentially unchanged especially if we use the mean-field approximation for the visible units which is what we do.

Two generative DBN models, one with and one without lateral connectivity, were trained using the updates from equations 2 and 4. The training data used consisted of 150,000 $20 \times 20$ patches extracted from images of natural scenes taken from the collection of Van Hateren[1]. The raw image intensities were pre-processed using a standard set of operations — namely an initial log-transformation, and then a normalisation step such that each pixel had zero-mean across the training set. The patches were then whitened using a Zero-Phase Components analysis (ZCA) filter-bank. The set of whitening filters is obtained by rotating the data into a co-ordinate system aligned with the eigenvectors of the covariance matrix, then rescaling each component by the inverse square-root of the corresponding eigenvalue, then rotating back into the original pixel co-ordinate system.

Using ZCA has a similar effect to learning lateral connections between pixels (Welling and Hinton, 2002). We used ZCA whitened data for both models to make it clear that the advantage of lateral connections is not just caused by their ability to whiten the input data. Because the data was whitened we did not include lateral connections in the bottom layer of the lateral DBN. The results presented in the figures that follow are all shown in "unwhitened pixel-space", i.e. the effects of the whitening filter are undone for display purposes.

The models each had 2000 units in the first hidden layer, 500 in the second hidden layer and 1000 units in the third hidden layer. The generative abilities of both models are very robust against variations in the number of hidden units in each layer, though it seems to be important for the top layer to be quite large. In the case where lateral connections were used, the first and second hidden layers of the final, composite model were fully laterally connected.

Data was taken in mini-batches of size 100, and training was performed for 50 epochs for the first layer and 30 epochs for the remaining layers. A learning rate of $10^{-3}$ was used for the interlayer connections, and half that rate for the lateral connections. Multiplicative weight decay of $10^{-2}$ multiplied by the learning rate was used, and a momentum factor of 0.9 was employed. When training the higher-level SRBM's in the model with lateral connectivity, 30 parallel mean field updates were used to produce the reconstructions with the top-down input from the hidden states held constant. Each mean field update set the new activity of every "visible" unit to be 0.2 times the previous activity plus 0.8 times the value computed by applying the logistic function to the total input received from the hidden units and the previous states of the visible units.

**Learned filters**

Figure 2 shows a random sample of the filters learned using an RBM with Gaussian visible units. These filters are the same for both models. This representation is $5\times$ overcomplete.

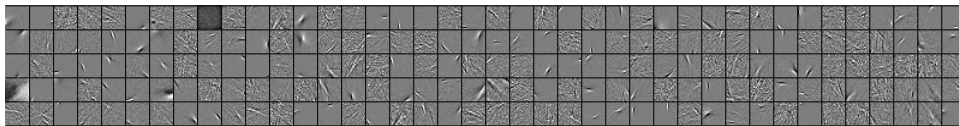

Figure 2: Filters from the first hidden layer. The results are generally similar to previous work on learning representations of natural image patches. The majority of the filters are tuned in location, orientation, and spatial frequency. The joint space of location and orientation is approximately evenly tiled and the spatial frequency responses span a range of about four octaves.

### 6.1.1 Generating samples from the model

The same issue that necessitates the use of approximations when learning deep-networks – namely the unknown value of the partition function – also makes it difficult to objectively assess how well they fit the data in the absence of predictive tasks such as classification. Since our main aim is to demonstrate the improvement in data modelling ability that lateral connections bring to DBN's, we simply present samples from similarly structured models, with and without lateral connections, and compare these samples with real data.

Ten-thousand data samples were generated by randomly initialising the top-level (S)RBM states and then running 300 iterations of a Gibbs sampling scheme between the top two layers. For models without lateral connections, each iteration of the scheme consisted of a full parallel-update of the top-most layer followed by a full parallel-update of the penultimate layer. In models with lateral connections, each iteration consisted of a full parallel-update of the top-most layer followed by 50 rounds of sequential stochastic updates of each unit in the penultimate layer, under the influence of the previously sampled top-layer states. (A different random ordering of units was drawn in each update-round.) After running this Markov Chain we then performed an ancestral generative pass down to the data layer. In the case of models with no lateral connections, this simply involved sampling from the factorial conditional distribution at each layer. In the case of models with lateral connections we performed 50 rounds of randomly-ordered, sequential stochastic updates under the influence of the top-down inputs from the layer above. In both cases, on the final hidden layer update before generating the pixel values, mean-field updates were used so that the data was generated using the real-valued probabilities in the first hidden layer rather than stochastic-binary states.

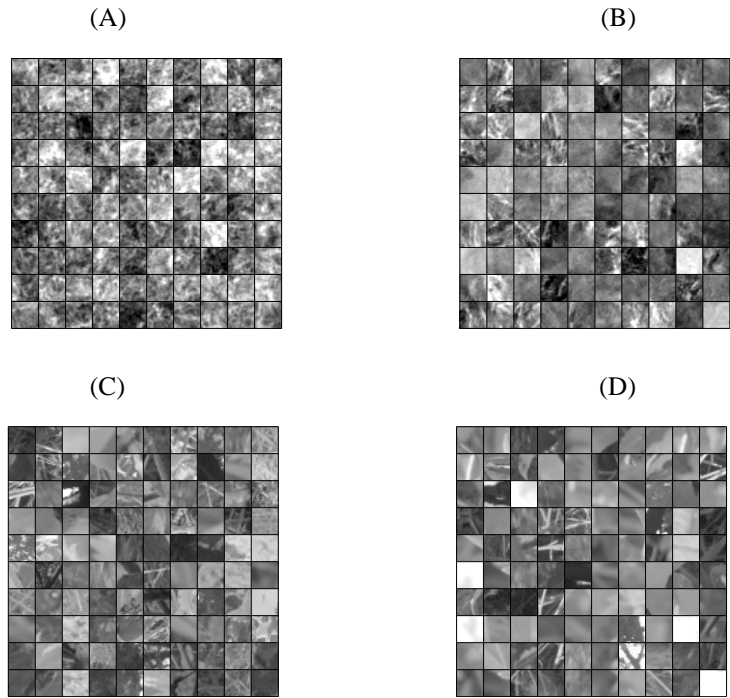

Figure 3: (A) Samples from a model without lateral connections. (B) Samples from a model with lateral connections. (C) Examples of actual data, drawn at random. (D) Examples of actual data, chosen to have closest cosine distance to samples from panel (B).

Figure 3 shows that adding undirected lateral interactions within each intermediate hidden layer of a deep belief net greatly improves the model's ability to generate image patches that look realistic. It is evident from the figure that the samples from the model with lateral connections are much more similar to the real data and contain much more coherent, long-range structure. Belief networks with only directed connections have difficulty capturing spatial constraints between the parts of an image because, in a directed network, the only way to enforce constraints is by using observed descendants. Unobserved ancestors can only be used to model shared sources of variation.

### 6.1.2 Marginal and pairwise statistics

In addition to the largely subjective comparisons from the previous section, if we perform some simple aggregate analyses of the synthesized data we see that the samples from the model with lateral connections are objectively well matched to those from true natural images. In the right-hand column of figure 4 we show histograms of pixel inensities for real data and for data generated by the

two models. The kurtosis is $8.3$ for real data, $7.3$ for the model with lateral connections, and $3.4$ for the model with no lateral connections. If we make a histogram of the outputs of all of the filters in the first hidden layer of the model, we discover that the kurtosis is $10.5$ on real data, $10.3$ on image patches generated by the model with lateral connections, and $3.8$ on patches generated by the other model.

Columns one through five of figure 4 show the distributions of the response of one filter conditional on the response of a second filter. Again, for image patches generated with lateral connections the statistics are similar to the data and without lateral connections they are quite different.

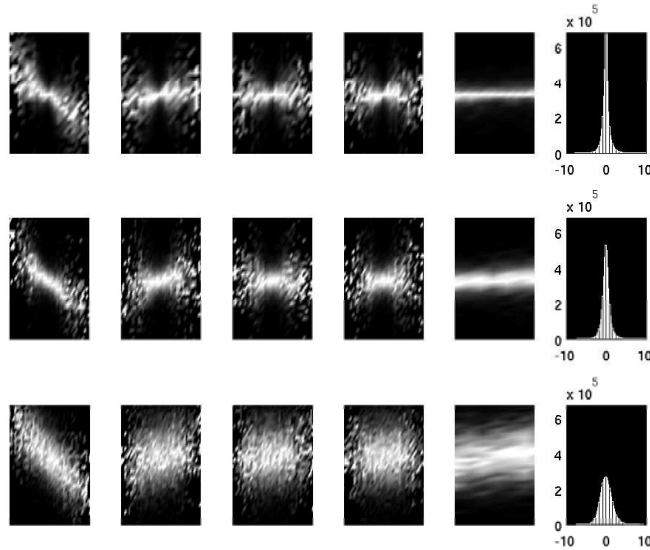

Figure 4: Each row shows statistics computed from a different set of 10,000 images. The first row is for real data. The second row is for image patches generated by the model with lateral interactions. The third row is for patches generated without lateral interactions. Column six shows histograms of pixel intensities. Columns 1-5 show conditional filter responses, in the style suggested in (Wainwright and Simoncelli, 2000), for two different gabor filters applied to the sampled images. In columns 1-3 the filters are 2, 4, or 8 pixels apart. In column 4 they are at the same location but orthogonal orientations. In column 5 they are at the same location and orientation but one octave apart in spatial frequency.

## 7    Discussion

Our results demonstrate the advantages of using semi-restricted Boltzmann machines as the building blocks when building deep belief nets. The model with lateral connections is very good at capturing the statistical structure of natural image patches. In future work we hope to exploit this in a number of image processing tasks that require a good prior over image patches.

The models presented in this paper had complete lateral connectivity — largely for simplicity in MATLAB. Such a strategy would not be feasible were we to significantly scale up our networks. Fortunately, there is an obvious solution to this — we can simply restrict the majority of lateral interactions to a local neighbourhood and concomittently have the hidden units focus their attention on spatially localised regions of the image. A topographic ordering would then exist throughout the various layers of the hierarchy. This would greatly reduce the computational load and it corresponds to a sensible prior over image structures, especially if the local regions get larger as we move up the hierarchy. Furthermore, it would probably make the process of settling within a layer much faster.

One limitation of the model we have described is that the top-down effects can only change the effective biases of the units in the Markov random field at each level. The model becomes much

more powerful if top-down effects can modulate the interactions. For example, an "edge" can be viewed as a breakdown in the local correlational structure of the image: pixel intensities cannot be predicted from neighbors on the other side of an object boundary. A hidden unit that can modulate the pairwise interactions rather than just the biases can form a far more abstract representation of an edge that is not tied to any particular contrast or intensity (Geman and Geman, 1984). Extending our model to this type of top-down modulation is fairly straightforward. Instead of using weights $w_{ij}$ that contribute energies $-v_i v_j w_{ij}$ we use weights $w_{ijk}$ that contribute energies $-v_i v_j h_k w_{ijk}$. This allows the binary state of $h_k$ to gate the effective weight between visible units $i$ and $j$. Memisevic and Hinton (2007) show that the same learning methods can be applied with a single hidden layer and there is no reason why such higher-order semi-restricted Boltzmann machines cannot be composed into deep belief nets.

Although we have focussed on the challenging task of modeling patches of natural images, we believe the ideas presented here are of much more general applicability. DBN's without lateral connections have produced state of the art results in a number of domains including document retrieval (Hinton and Salakhutdinov, 2006), character recognition (Hinton et al., 2006), lossy image compression (Hinton and Salakhutdinov, 2006), and the generation of human motion (Taylor et al., 2007). Lateral connections may help in all of these domains.

**Acknowledgments**

We are grateful to the members of the machine learning group at the University of Toronto for helpful discussions. This work was supported by NSERC, CFI and CIFAR. GEH is a fellow of CIFAR and holds a CRC chair.

## Footnotes

[1]http://hlab.phys.rug.nl/imlib/index.html

# References

Bell, A. J. and Sejnowski, T. J. (1997). The "independent components" of natural scenes are edge filters. *Vision Research*, 37(23):3327–3338.

Bengio, Y., Lamblin, P., Popovici, D., and Larochelle, H. (2007). Greedy layer-wise training of deep networks. In B., S., Platt, J., and Hoffman, T., editors, *Advances in Neural Information Processing Systems 19*. MIT Press, Cambridge, MA.

Geman, S. and Geman, D. (1984). Stochastic relaxation, gibbs distributions and the bayesian restoration of images. *IEEE Trans. Pattern Anal. Mach. Intell*, 6.

Hinton, G. E. (2002). Training products of experts by minimizing contrastive divergence. *Neural Computation*, 14(8):1711–1800.

Hinton, G. E., Osindero, S., and Teh, Y. W. (2006). A fast learning algorithm for deep belief nets. *Neural Computation*, 18.

Hinton, G. E. and Salakhutdinov, R. (2006). Reducing the dimensionality of data with neural networks. *Science*, 313.

Karklin, Y. and Lewicki, M. (2005). A hierarchical bayesian model for learning nonlinear statistical regularities in nonstationary natural signals. *Neural Computation*, 17(2).

Lyu, S. and Simoncelli, E. (2006). Statistical modeling of images with fields of gaussian scale mixtures. In *Advances Neural Information Processing Systems*, volume 19.

Memisevic, R. F. and Hinton, G. E. (2007). Unsupervised learning of image transformations. In *Computer Vision and Pattern Recognition*. IEEE Computer Society.

Olshausen, B. A. and Field, D. J. (1996). Emergence of simple-cell receptive field properties by learning a sparse code for natural images. *Nature*, 381(6583):607–609. JUN 13 NATURE.

Osindero, S., Welling, M., and Hinton, G. E. (2006). Topographic product models applied to natural scene statistics. *Neural Computation*, 18(2).

Taylor, G. W., Hinton, G. E., and Roweis, S. (2007). Modeling human motion using binary latent variables. In B., S., Platt, J., and Hoffman, T., editors, *Advances in Neural Information Processing Systems 19*. MIT Press, Cambridge, MA.

Wainwright, M. and Simoncelli, E. (2000). Scale mixtures of Gaussians and the statistics of natural images. In *Advances Neural Information Processing Systems*, volume 12, pages 855–861.

Welling, M. and Hinton, G. E. (2002). A new learning algorithm for mean field boltzmann machines. In *International Joint Conference on Neural Networks*, Madrid.

